# Statistical Performance of Convex Tensor Decomposition

**Ryota Tomioka**[†]     **Taiji Suzuki**[†]
[†]Department of Mathematical Informatics,
The University of Tokyo
Tokyo 113-8656, Japan
tomioka@mist.i.u-tokyo.ac.jp
s-taiji@stat.t.u-tokyo.ac.jp

**Kohei Hayashi**[‡]
[‡]Graduate School of Information Science,
Nara Institute of Science and Technology
Nara 630-0192, Japan
kohei-h@is.naist.jp

**Hisashi Kashima**[†,*]
[*]Basic Research Programs PRESTO,
Synthesis of Knowledge for Information Oriented Society, JST
Tokyo 102-8666, Japan
kashima@mist.i.u-tokyo.ac.jp

## Abstract

We analyze the statistical performance of a recently proposed convex tensor decomposition algorithm. Conventionally tensor decomposition has been formulated as non-convex optimization problems, which hindered the analysis of their performance. We show under some conditions that the mean squared error of the convex method scales linearly with the quantity we call the normalized rank of the true tensor. The current analysis naturally extends the analysis of convex low-rank matrix estimation to tensors. Furthermore, we show through numerical experiments that our theory can precisely predict the scaling behaviour in practice.

## 1   Introduction

Tensors (multi-way arrays) generalize matrices and naturally represent data having more than two modalities. For example, multi-variate time-series, for instance, electroencephalography (EEG), recorded from multiple subjects under various conditions naturally form a tensor. Moreover, in collaborative filtering, users' preferences on products, conventionally represented as a matrix, can be represented as a *tensor* when the preferences change over time or context.

For the analysis of tensor data, various models and methods for the low-rank decomposition of tensors have been proposed (see Kolda & Bader [12] for a recent survey). These techniques have recently become increasingly popular in data-mining [1, 14] and computer vision [25, 26]. Besides they have proven useful in chemometrics [4], psychometrics [24], and signal processing [20, 7, 8].

Despite empirical success, the statistical performance of tensor decomposition *algorithms* has not been fully elucidated. The difficulty lies in the *non-convexity* of the conventional tensor decomposition algorithms (e.g., alternating least squares [6]). In addition, studies have revealed many discrepancies (see [12]) between matrix rank and tensor rank, which make extension of studies on the performance of low-rank matrix models (e.g., [9]) challenging.

Recently, several authors [21, 10, 13, 23] have focused on the notion of *tensor mode-k rank* (instead of tensor rank), which is related to the Tucker decomposition [24]. They discovered that regularized estimation based on the Schatten 1-norm, which is a popular technique for recovering low-rank matrices via convex optimization, can also be applied to tensor decomposition. In particular, the

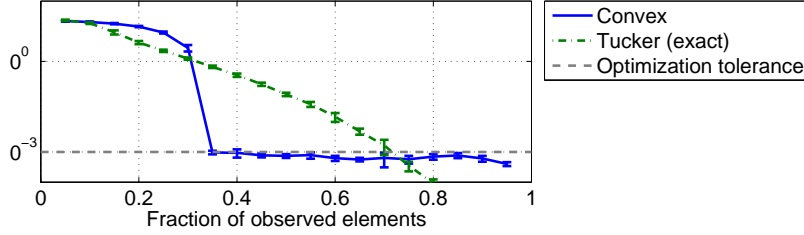

Figure 1: Result of estimation of rank-$(7, 8, 9)$ tensor of dimensions $50 \times 50 \times 20$ from partial measurements; see [23] for the details. The estimation error $\left\| \hat{\mathcal{W}} - \mathcal{W}^* \right\|_F$ is plotted against the fraction of observed elements $m = M/N$. Error bars over 10 repetitions are also shown. Convex refers to the convex tensor decomposition based on the minimization problem (7). Tucker (exact) refers to the conventional (non-convex) Tucker decomposition [24] at the correct rank. Gray dashed line shows the optimization tolerance $10^{-3}$. The question is how we can predict the point where the generalization begins (roughly $m = 0.35$ in this plot).

study in [23] showed that there is a clear transition at certain number of samples where the error drops dramatically from no generalization to perfect generalization (see Figure 1).

In this paper, motivated by the above recent work, we mathematically analyze the performance of convex tensor decomposition. The new convex formulation for tensor decomposition allows us to generalize recent results on Schatten 1-norm-regularized estimation of matrices (see [17, 18, 5, 19]). Under a general setting we show how the estimation error scales with the mode-$k$ ranks of the true tensor. Furthermore, we analyze the specific settings of (i) noisy tensor decomposition and (ii) random Gaussian design. In the first setting, we assume that all the elements of a low-rank tensor is observed with noise and the goal is to recover the underlying low-rank structure. This is the most common setting a tensor decomposition algorithm is used. In the second setting, we assume that the unknown tensor is a coefficient of a tensor-input scalar-output regression problem and the input tensors (design) are randomly given from independent Gaussian distributions. Surprisingly, it turns out that the random Gaussian setting can precisely predict the phase-transition-like behaviour in Figure 1. To the best of our knowledge, this is the first paper that rigorously studies the performance of a tensor decomposition algorithm.

## 2 Notation

In this section, we introduce the notations we use in this paper. Moreover, we introduce a Hölder-like inequality (3) and the notion of mode-$k$ decomposability (5), which play central roles in our analysis.

Let $\mathcal{X} \in \mathbb{R}^{n_1 \times \cdots n_K}$ be a $K$-way tensor. We denote the number of elements in $\mathcal{X}$ by $N = \prod_{k=1}^{K} n_k$. The inner product between two tensors $\langle \mathcal{W}, \mathcal{X} \rangle$ is defined as $\langle \mathcal{W}, \mathcal{X} \rangle = \text{vec}(\mathcal{W})^\top \text{vec}(\mathcal{X})$, where vec is a vectorization. In addition, we define the Frobenius norm of a tensor $\left\| \mathcal{X} \right\|_F = \sqrt{\langle \mathcal{X}, \mathcal{X} \rangle}$.

The *mode-$k$ unfolding* $\boldsymbol{X}_{(k)}$ is the $n_k \times \bar{n}_{\backslash k}$ ($\bar{n}_{\backslash k} := \prod_{k' \neq k} n_{k'}$) matrix obtained by concatenating the mode-$k$ fibers (the vectors obtained by fixing every index of $\mathcal{X}$ but the $k$th index) of $\mathcal{X}$ as column vectors. The mode-$k$ rank of a tensor $\mathcal{X}$, denoted by $\text{rank}_k(\mathcal{X})$, is the rank of the mode-$k$ unfolding $\boldsymbol{X}_{(k)}$ (as a matrix). Note that when $K = 2$ and $\mathcal{X}$ is actually a matrix, and $\boldsymbol{X}_{(2)} = \boldsymbol{X}_{(1)}^\top$. We say a tensor $\mathcal{X}$ is rank $(r_1, \ldots, r_K)$ when $r_k = \text{rank}_k(\mathcal{X})$ for $k = 1, \ldots, K$. Note that the mode-$k$ rank can be computed in a polynomial time, because it boils down to computing a matrix rank, whereas computing tensor rank is NP complete [11]. See [12] for more details.

Since for each $k$, the convex envelope of the mode-$k$ rank is given as the Schatten 1-norm [18] (known as the trace norm [22] or the nuclear norm [3]), it is natural to consider the following

overlapped Schatten 1-norm $\left\|\!\left\|\mathcal{W}\right\|\!\right\|_{S_1}$ of a tensor $\mathcal{W} \in \mathbb{R}^{n_1 \times \cdots \times n_K}$ (see also [21]):

$$\left\|\!\left\|\mathcal{W}\right\|\!\right\|_{S_1} = \frac{1}{K} \sum_{k=1}^{K} \left\|\boldsymbol{W}_{(k)}\right\|_{S_1},\tag{1}$$

where $\boldsymbol{W}_{(k)}$ is the mode-$k$ unfolding of $\mathcal{W}$. Here $\|\cdot\|_{S_1}$ is the Schatten 1-norm for a matrix

$$\|\boldsymbol{W}\|_{S_1} = \sum\nolimits_{j=1}^{r} \sigma_j(\boldsymbol{W}),$$

where $\sigma_j(\boldsymbol{W})$ is the $j$th largest singular-value of $\boldsymbol{W}$. The dual norm of the Schatten 1-norm is the Schatten $\infty$-norm (known as the spectral norm) as follows:

$$\|\boldsymbol{X}\|_{S_\infty} = \max_{j=1,\dots,r} \sigma_j(\boldsymbol{X}).$$

Since the two norms $\|\cdot\|_{S_1}$ and $\|\cdot\|_{S_\infty}$ are dual to each other, we have the following inequality:

$$|\langle \boldsymbol{W}, \boldsymbol{X}\rangle| \le \|\boldsymbol{W}\|_{S_1}\|\boldsymbol{X}\|_{S_\infty},\tag{2}$$

where $\langle \boldsymbol{W}, \boldsymbol{X}\rangle$ is the inner product of $\boldsymbol{W}$ and $\boldsymbol{X}$.

The same inequality holds for the overlapped Schatten 1-norm (1) and its dual norm. The dual norm of the overlapped Schatten 1-norm can be characterized by the following lemma.

**Lemma 1.** *The dual norm of the overlapped Schatten 1-norm denoted as $\left\|\!\left\|\cdot\right\|\!\right\|_{S_1^*}$ is defined as the infimum of the maximum mode-$k$ spectral norm over the tensors whose average equals the given tensor $\mathcal{X}$ as follows:*

$$\left\|\!\left\|\mathcal{X}\right\|\!\right\|_{S_1^*} = \inf_{\frac{1}{K}\left(\mathcal{Y}^{(1)}+\mathcal{Y}^{(2)}+\cdots+\mathcal{Y}^{(K)}\right)=\mathcal{X}} \max_{k=1,\dots,K} \|\boldsymbol{Y}_{(k)}^{(k)}\|_{S_\infty},$$

*where $\boldsymbol{Y}_{(k)}^{(k)}$ is the mode-$k$ unfolding of $\mathcal{Y}^{(k)}$. Moreover, the following upper bound on the dual norm $\left\|\!\left\|\cdot\right\|\!\right\|_{S_1^*}$ is valid:*

$$\left\|\!\left\|\mathcal{X}\right\|\!\right\|_{S_1^*} \le \left\|\!\left\|\mathcal{X}\right\|\!\right\|_{\mathrm{mean}} := \frac{1}{K} \sum\nolimits_{k=1}^{K} \|\boldsymbol{X}_{(k)}\|_{S_\infty}.$$

*Proof.* The first part can be shown by solving the dual of the maximization problem $\left\|\!\left\|\mathcal{X}\right\|\!\right\|_{S_1^*} := \sup \langle \mathcal{W}, \mathcal{X}\rangle$ s.t. $\left\|\!\left\|\mathcal{W}\right\|\!\right\|_{S_1} \le 1$. The second part is obtained by setting $\mathcal{Y}^{(k)} = \frac{K}{\sum_{k'=1}^{K} 1/c_{k'}} \mathcal{X}/c_k$, where $c_k = \|\boldsymbol{X}_{(k)}\|_{S_\infty}$, and using Jensen's inequality. $\qquad\square$

According to Lemma 1, we have the following Hölder-like inequality

$$|\langle \mathcal{W}, \mathcal{X}\rangle| \le \left\|\!\left\|\mathcal{W}\right\|\!\right\|_{S_1}\left\|\!\left\|\mathcal{X}\right\|\!\right\|_{S_1^*} \le \left\|\!\left\|\mathcal{W}\right\|\!\right\|_{S_1}\left\|\!\left\|\mathcal{X}\right\|\!\right\|_{\mathrm{mean}}.\tag{3}$$

Note that the above bound is *tighter* than the more intuitive relation $|\langle \mathcal{W}, \mathcal{X}\rangle| \le \left\|\!\left\|\mathcal{W}\right\|\!\right\|_{S_1}\left\|\!\left\|\mathcal{X}\right\|\!\right\|_{S_\infty}$ ($\left\|\!\left\|\mathcal{X}\right\|\!\right\|_{S_\infty} := \max_{1,\dots,K} \|\boldsymbol{X}_{(k)}\|_{S_\infty}$), which one might come up as an analogy to the matrix case (2).

Finally, let $\mathcal{W}^* \in \mathbb{R}^{n_1 \times \cdots \times n_K}$ be the low-rank tensor that we wish to recover. We assume that $\mathcal{W}^*$ is rank $(r_1, \dots, r_K)$. Thus, for each $k$ we have

$$\boldsymbol{W}_{(k)}^* = \boldsymbol{U}_k \boldsymbol{S}_k \boldsymbol{V}_k \qquad (k = 1, \dots, K),$$

where $\boldsymbol{U}_k \in \mathbb{R}^{n_k \times r_k}$ and $\boldsymbol{V}_k \in \mathbb{R}^{\bar{n}_{\backslash k} \times r_k}$ are orthogonal, and $\boldsymbol{S}_k \in \mathbb{R}^{r_k \times r_k}$ is diagonal. Let $\Delta \in \mathbb{R}^{n_1 \times \cdots \times n_K}$ be an arbitrary tensor. We define the mode-$k$ orthogonal complement $\boldsymbol{\Delta}_k''$ of an unfolding $\boldsymbol{\Delta}_{(k)} \in \mathbb{R}^{n_k \times \bar{n}_{\backslash k}}$ of $\Delta$ with respect to the true low-rank tensor $\mathcal{W}^*$ as follows:

$$\boldsymbol{\Delta}_k'' = (\boldsymbol{I}_{n_k} - \boldsymbol{U}_k\boldsymbol{U}_k^\top)\boldsymbol{\Delta}_{(k)}(\boldsymbol{I}_{\bar{n}_{\backslash k}} - \boldsymbol{V}_k\boldsymbol{V}_k^\top).\tag{4}$$

In addition $\boldsymbol{\Delta}_k' := \boldsymbol{\Delta}_{(k)} - \boldsymbol{\Delta}_k''$ is the component having overlapped row/column space with the unfolding of the true tensor $\boldsymbol{W}_{(k)}^*$. Note that the decomposition $\boldsymbol{\Delta}_{(k)} = \boldsymbol{\Delta}_k' + \boldsymbol{\Delta}_k''$ is defined for each mode; thus we use subscript $k$ instead of $(k)$.

Using the decomposition defined above we have the following equality, which we call *mode-$k$ decomposability* of the Schatten 1-norm:

$$\|\boldsymbol{W}_{(k)}^* + \boldsymbol{\Delta}_k''\|_{S_1} = \|\boldsymbol{W}_{(k)}^*\|_{S_1} + \|\boldsymbol{\Delta}_k''\|_{S_1} \quad (k = 1, \dots, K).\tag{5}$$

The above decomposition is defined for each mode and thus it is weaker than the notion of decomposability discussed by Negahban et al. [15].

## 3 Theory

In this section, we first present a deterministic result that holds under a certain choice of regularization constant $\lambda_M$ and an assumption called the restricted strong convexity. Then, we focus on special cases to justify the choice of regularization constant and the restricted strong convexity assumption. We analyze the setting of (i) noisy tensor decomposition and (ii) random Gaussian design in Section 3.2 and Section 3.3, respectively.

### 3.1 Main result

Our goal is to estimate an unknown rank $(r_1, \ldots, r_K)$ tensor $\mathcal{W}^* \in \mathbb{R}^{n_1 \times \cdots n_K}$ from observations

$$y_i = \langle \mathcal{X}_i, \mathcal{W}^* \rangle + \epsilon_i \quad (i = 1, \ldots, M). \tag{6}$$

Here the noise $\epsilon_i$ follows the independent zero-mean Gaussian distribution with variance $\sigma^2$.

We employ the regularized empirical risk minimization problem proposed in [21, 10, 13, 23] for the estimation of $\mathcal{W}$ as follows:

$$\operatorname*{minimize}_{\mathcal{W} \in \mathbb{R}^{n_1 \times \cdots \times n_K}} \quad \frac{1}{2M} \|\boldsymbol{y} - \mathfrak{X}(\mathcal{W})\|_2^2 + \lambda_M \|\|\mathcal{W}\|\|_{S_1}, \tag{7}$$

where $\boldsymbol{y} = (y_1, \ldots, y_M)^\top$ is the collection of observations; $\mathfrak{X} : \mathbb{R}^{n_1 \times \cdots \times n_K} \to \mathbb{R}^M$ is a linear operator that maps $\mathcal{W}$ to the $M$ dimensional output vector $\mathfrak{X}(\mathcal{W}) = (\langle \mathcal{X}_1, \mathcal{W} \rangle, \ldots, \langle \mathcal{X}_M, \mathcal{W} \rangle)^\top \in \mathbb{R}^M$. The Schatten 1-norm term penalizes every mode of $\mathcal{W}$ to be jointly low-rank (see Equation (1)); $\lambda_M > 0$ is the regularization constant. Accordingly, the solution of the minimization problem (7) is typically a low-rank tensor when $\lambda_M$ is sufficiently large. In addition, we denote the adjoint operator of $\mathfrak{X}$ as $\mathfrak{X}^* : \mathbb{R}^M \to \mathbb{R}^{n_1 \times \cdots \times n_K}$; that is $\mathfrak{X}^*(\boldsymbol{\epsilon}) = \sum_{i=1}^M \epsilon_i \mathcal{X}_i \in \mathbb{R}^{n_1 \times \cdots \times n_K}$.

The first step in our analysis is to characterize the particularity of the residual tensor $\Delta := \hat{\mathcal{W}} - \mathcal{W}^*$ as in the following lemma.

**Lemma 2.** *Let $\hat{\mathcal{W}}$ be the solution of the minimization problem* (7) *with $\lambda_M \geq 2\|\|\mathfrak{X}^*(\boldsymbol{\epsilon})\|\|_{\mathrm{mean}}/M$, and let $\Delta := \hat{\mathcal{W}} - \mathcal{W}^*$, where $\mathcal{W}^*$ is the true low-rank tensor. Let $\boldsymbol{\Delta}_{(k)} = \boldsymbol{\Delta}_k' + \boldsymbol{\Delta}_k''$ be the decomposition defined in Equation* (4). *Then we have the following inequalities:*

1. $\operatorname{rank}(\boldsymbol{\Delta}_k') \leq 2r_k$ *for each $k = 1, \ldots, K$.*

2. $\sum_{k=1}^K \|\boldsymbol{\Delta}_k''\|_{S_1} \leq 3 \sum_{k=1}^K \|\boldsymbol{\Delta}_k'\|_{S_1}$.

*Proof.* The proof uses the mode-$k$ decomposability (5) and is analogous to that of Lemma 1 in [17]. $\square$

The second ingredient of our analysis is the restricted strong convexity. Although, "strong" may sound like a strong assumption, the point is that we require this assumption to hold only for the particular residual tensor we characterized in Lemma 2. The assumption can be stated as follows.

**Assumption 1** (Restricted strong convexity)**.** *We suppose that there is a positive constant $\kappa(\mathfrak{X})$ such that the operator $\mathfrak{X}$ satisfies the inequality*

$$\frac{1}{M} \|\mathfrak{X}(\Delta)\|_2^2 \geq \kappa(\mathfrak{X}) \|\|\Delta\|\|_F^2, \tag{8}$$

*for all $\Delta \in \mathbb{R}^{n_1 \times \cdots \times n_K}$ such that for each $k = 1, \ldots, K$, $\operatorname{rank}(\boldsymbol{\Delta}_k') \leq 2r_k$ and $\sum_{k=1}^K \|\boldsymbol{\Delta}_k''\|_{S_1} \leq 3 \sum_{k=1}^K \|\boldsymbol{\Delta}_k'\|_{S_1}$, where $\boldsymbol{\Delta}_k'$ and $\boldsymbol{\Delta}_k''$ are defined through the decomposition* (4).

Now using the above two ingredients, we are ready to prove the following deterministic guarantee on the performance of the estimation procedure (7).

**Theorem 1.** *Let $\hat{\mathcal{W}}$ be the solution of the minimization problem* (7) *with $\lambda_M \geq 2\|\|\mathfrak{X}^*(\boldsymbol{\epsilon})\|\|_{\mathrm{mean}}/M$. Suppose that the operator $\mathfrak{X}$ satisfies the restricted strong convexity condition. Then the following bound is true:*

$$\|\|\hat{\mathcal{W}} - \mathcal{W}^*\|\|_F \leq \frac{32\lambda_M \sum_{k=1}^K \sqrt{r_k}}{\kappa(\mathfrak{X})K}. \tag{9}$$

*Proof.* Let $\Delta = \hat{\mathcal{W}} - \mathcal{W}^*$. Combining the fact that the objective value for $\hat{\mathcal{W}}$ is smaller than that for $\mathcal{W}^*$, the Hölder-like inequality (3), the triangular inequality $\left\|\left\|\mathcal{W}^*\right\|\right\|_{S_1} - \left\|\left\|\hat{\mathcal{W}}\right\|\right\|_{S_1} \leq \left\|\left\|\Delta\right\|\right\|_{S_1}$, and the assumption $\left\|\left\|\mathfrak{X}^*(\epsilon)/M\right\|\right\|_{\mathrm{mean}} \leq \lambda_M/2$, we obtain

$$\frac{1}{2M}\|\mathfrak{X}(\Delta)\|_2^2 \leq \left\|\left\|\mathfrak{X}^*(\epsilon)/M\right\|\right\|_{\mathrm{mean}}\left\|\left\|\Delta\right\|\right\|_{S_1} + \lambda_M\left\|\left\|\Delta\right\|\right\|_{S_1} \leq 2\lambda_M\left\|\left\|\Delta\right\|\right\|_{S_1}. \tag{10}$$

Now the left-hand side can be lower-bounded using the restricted strong convexity (8). On the other hand, using Lemma 2, the right-hand side can be upper-bounded as follows:

$$\left\|\left\|\Delta\right\|\right\|_{S_1} \leq \tfrac{1}{K}\sum_{k=1}^K(\|\boldsymbol{\Delta}_k'\|_{S_1} + \|\boldsymbol{\Delta}_k''\|_{S_1}) \leq \tfrac{4}{K}\sum_{k=1}^K\|\boldsymbol{\Delta}_k'\|_{S_1} \leq \tfrac{4\left\|\left\|\Delta\right\|\right\|_F}{K}\sum_{k=1}^K\sqrt{2r_k}, \tag{11}$$

where the last inequality follows because $\left\|\left\|\Delta\right\|\right\|_F = \|\boldsymbol{\Delta}_{(k)}\|_F$ for $k = 1, \ldots, K$. Combining inequalities (8), (10), and (11), we obtain our claim (9). $\qquad\square$

Negahban et al. [15] (see also [17]) pointed out that the key properties for establishing a sharp convergence result for a regularized $M$-estimator is the decomposability of the regularizer and the restricted strong convexity. What we have shown suggests that the weaker mode-$k$ decomposability (5) suffice to obtain the above convergence result for the overlapped Schatten 1-norm (1) regularization.

## 3.2 Noisy Tensor Decomposition

In this subsection, we consider the setting where all the elements are observed (with noise) and the goal is to recover the underlying low-rank tensor without noise.

Since all the elements are observed only once, $\mathfrak{X}$ is simply a vectorization ($M = N$), and the left-hand side of inequality (10) gives the quantity of interest $\|\mathfrak{X}(\Delta)\|_2^2 = \left\|\left\|\hat{\mathcal{W}} - \mathcal{W}^*\right\|\right\|_F$. Therefore, the remaining task is to bound $\left\|\left\|\mathfrak{X}^*(\epsilon)\right\|\right\|_{\mathrm{mean}}$ as in the following lemma.

**Lemma 3.** *Suppose that $\mathfrak{X} : n_1 \times \cdots \times n_K \to N$ is a vectorization of a tensor. With high probability the quantity $\left\|\left\|\mathfrak{X}^*(\epsilon)\right\|\right\|_{\mathrm{mean}}$ is concentrated around its mean, which can be bounded as follows:*

$$\mathbb{E}\left\|\left\|\mathfrak{X}^*(\epsilon)\right\|\right\|_{\mathrm{mean}} \leq \frac{\sigma}{K}\sum_{k=1}^K\left(\sqrt{n_k} + \sqrt{\bar{n}_{\backslash k}}\right). \tag{12}$$

Setting the regularization constant as $\lambda_M = c_0\mathbb{E}\left\|\left\|\mathfrak{X}^*(\epsilon)\right\|\right\|_{\mathrm{mean}}/N$, we obtain the following theorem.

**Theorem 2.** *Suppose that $\mathfrak{X} : n_1 \times \cdots \times n_K \to N$ is a vectorization of a tensor. There are universal constants $c_0$ and $c_1$, such that, with high probability, any solution of the minimization problem (7) with regularization constant $\lambda_M = c_0\sigma\sum_{k=1}^K(\sqrt{n_k} + \sqrt{\bar{n}_{\backslash k}})/(KN)$ satisfies the following bound:*

$$\left\|\left\|\hat{\mathcal{W}} - \mathcal{W}^*\right\|\right\|_F^2 \leq c_1\sigma^2\left(\frac{1}{K}\sum_{k=1}^K\left(\sqrt{n_k} + \sqrt{\bar{n}_{\backslash k}}\right)\right)^2\left(\frac{1}{K}\sum_{k=1}^K\sqrt{r_k}\right)^2.$$

*Proof.* Combining Equations (10)–(11) with the fact that $\mathfrak{X}$ is simply a vectorization and $M = N$, we have

$$\tfrac{1}{N}\|\hat{\mathcal{W}} - \mathcal{W}^*\|_F \leq \tfrac{16\sqrt{2}\lambda_M}{K}\sum_{k=1}^K\sqrt{r_k}.$$

Substituting the choice of regularization constant $\lambda_M$ and squaring both sides, we obtain our claim. $\square$

We can simplify the result of Theorem 2 by noting that $\bar{n}_{\backslash k} = N/n_k \gg n_k$, when the dimensions are of the same order. Introducing the notation $\|\boldsymbol{r}\|_{1/2} = (\frac{1}{K}\sum_{k=1}^K\sqrt{r_k})^2$ and $\boldsymbol{n}^{-1} := (1/n_1, \ldots, 1/n_K)$, we have

$$\frac{\left\|\left\|\hat{\mathcal{W}} - \mathcal{W}^*\right\|\right\|_F^2}{N} \leq O_p\left(\sigma^2\|\boldsymbol{n}^{-1}\|_{1/2}\|\boldsymbol{r}\|_{1/2}\right). \tag{13}$$

We call the quantity $\bar{r} = \|\boldsymbol{n}^{-1}\|_{1/2}\|\boldsymbol{r}\|_{1/2}$ the *normalized rank*, because $\bar{r} = r/n$ when the dimensions are balanced ($n_k = n$ and $r_k = r$ for all $k = 1, \ldots, K$).

### 3.3 Random Gaussian Design

In this subsection, we consider the case the elements of the input tensors $\mathcal{X}_i$ ($i = 1, \ldots, M$) in the observation model (6) are distributed according to independent identical standard Gaussian distributions. We call this setting *random Gaussian design*.

First we show an upper bound on the norm $\left\|\left\|\left\|\mathfrak{X}^*(\epsilon)\right\|\right\|\right\|_{\text{mean}}$, which we use to specify the scaling of the regularization constant $\lambda_M$ in Theorem 1.

**Lemma 4.** *Let $\mathfrak{X} : \mathbb{R}^{n_1 \times \cdots \times n_K} \to \mathbb{R}^M$ be a random Gaussian design. In addition, we assume that the noise $\epsilon_i$ is sampled independently from $\mathcal{N}(0, \sigma^2)$. Then with high probability the quantity $\left\|\left\|\left\|\mathfrak{X}^*(\epsilon)\right\|\right\|\right\|_{\text{mean}}$ is concentrated around its mean, which can be bounded as follows:*

$$\mathbb{E}\left\|\left\|\left\|\mathfrak{X}^*(\epsilon)\right\|\right\|\right\|_{\text{mean}} \leq \frac{\sigma\sqrt{M}}{K} \sum_{k=1}^{K} \left(\sqrt{n_k} + \sqrt{\bar{n}_{\backslash k}}\right).$$

Next the following lemma, which is a generalization of a result presented in Negahban and Wainwright [17, Proposition 1], provides a ground for the restricted strong convexity assumption (8).

**Lemma 5.** *Let $\mathfrak{X} : \mathbb{R}^{n_1 \times \cdots \times n_K} \to \mathbb{R}^M$ be a random Gaussian design. Then it satisfies*

$$\frac{\|\mathfrak{X}(\Delta)\|_2}{\sqrt{M}} \geq \frac{1}{4}\left\|\left\|\left\|\Delta\right\|\right\|\right\|_F - \frac{1}{K}\sum_{k=1}^{K}\left(\sqrt{\frac{n_k}{M}} + \sqrt{\frac{\bar{n}_{\backslash k}}{M}}\right)\left\|\left\|\left\|\Delta\right\|\right\|\right\|_{S_1},$$

*with probability at least $1 - 2\exp(-N/32)$.*

*Proof.* The proof is analogous to that of Proposition 1 in [17] except that we use Hölder-like inequality (3) for tensors instead of inequality (2) for matrices. □

Finally, we obtain the following convergence bound.

**Theorem 3.** *Under the random Gaussian design setup, there are universal constants $c_0$, $c_1$, and $c_2$ such that for a sample size $M \geq c_1(\frac{1}{K}\sum_{k=1}^{K}(\sqrt{n_k} + \sqrt{\bar{n}_{\backslash k}}))^2(\frac{1}{K}\sum_{k=1}^{K}\sqrt{r_k})^2$, any solution of the minimization problem (7) with regularization constant $\lambda_M = c_0\sigma\sum_{k=1}^{K}(\sqrt{n_k} + \sqrt{\bar{n}_{\backslash k}})/(K\sqrt{M})$ satisfies the following bound:*

$$\left\|\left\|\left\|\hat{\mathcal{W}} - \mathcal{W}^*\right\|\right\|\right\|_F^2 \leq c_2 \frac{\sigma^2(\frac{1}{K}\sum_{k=1}^{K}(\sqrt{n_k} + \sqrt{\bar{n}_{\backslash k}}))^2(\frac{1}{K}\sum_{k=1}^{K}\sqrt{r_k})^2}{M},$$

*with high probability.*

Again we can simplify the result of Theorem 3 as follows: for sample size $M \geq c_1 N\bar{r}$ we have

$$\left\|\left\|\left\|\hat{\mathcal{W}} - \mathcal{W}^*\right\|\right\|\right\|_F^2 \leq O_p\left(\sigma^2 \frac{N\|\boldsymbol{n}^{-1}\|_{1/2}\|\boldsymbol{r}\|_{1/2}}{M}\right), \tag{14}$$

where $\bar{r} = \|\boldsymbol{n}^{-1}\|_{1/2}\|\boldsymbol{r}\|_{1/2}$ is the normalized rank. Note that the condition on the number of samples $M$ does *not* depend on the noise variance $\sigma^2$. Therefore in the limit $\sigma^2 \to 0$, the bound (14) is sufficiently small but only valid for sample size $M$ that exceeds $c_1 N\bar{r}$, which implies a threshold behavior as in Figure 1.

Note also that in the matrix case ($K = 2$), $r_1 = r_2 = r$ and $N\|\boldsymbol{n}^{-1}\|_{1/2} = O(n_1 + n_2)$. Therefore we can restate the above result as for sample size $M \geq c_1 r(n_1 + n_2)$, we have $\|\hat{\boldsymbol{W}} - \boldsymbol{W}^*\|_F^2 \leq O_p(r(n_1 + n_2)/M)$, which is compatible with the result in [17, 18].

## 4 Experiments

In this section, we conduct two numerical experiments to confirm our analysis in Section 3.2 and Section 3.3.

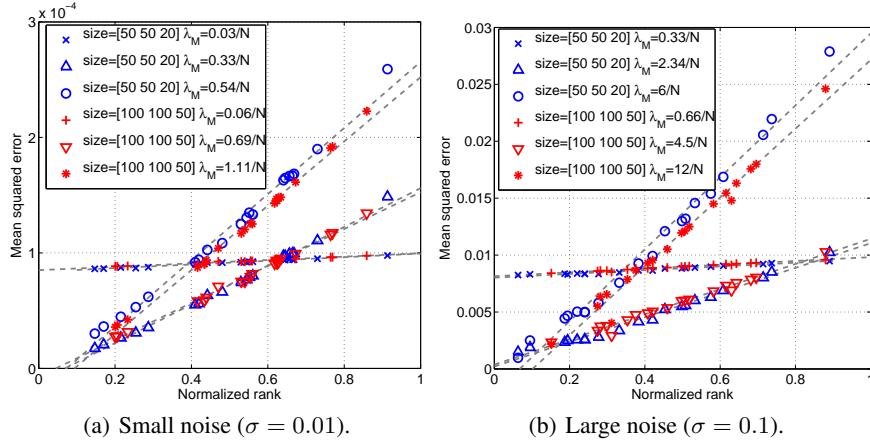

(a) Small noise ($\sigma = 0.01$).　　　　(b) Large noise ($\sigma = 0.1$).

Figure 2: Result of noisy tensor decomposition for tensors of size $50 \times 50 \times 20$ and $100 \times 100 \times 50$.

## 4.1 Noisy Tensor Decomposition

We randomly generated low-rank tensors of dimensions $\boldsymbol{n}^{(1)} = (50, 50, 20)$ and $\boldsymbol{n}^{(2)} = (100, 100, 50)$ for various ranks $(r_1, \ldots, r_K)$. For a specific rank, we generated the true tensor by drawing elements of the $r_1 \times \cdots \times r_K$ "core tensor" from the standard normal distribution and multiplying its each mode by an orthonormal factor randomly drawn from the Haar measure. As described in Section 3.2, the observation $\boldsymbol{y}$ consists of all the elements of the original tensor once ($M = N$) with additive independent Gaussian noise with variance $\sigma^2$. We used the alternating direction method of multipliers (ADMM) for "constraint" approaches described in [23, 10] to solve the minimization problem (7). The whole experiment was repeated 10 times and averaged.

The results are shown in Figure 2. The mean squared error $\left\| \hat{\mathcal{W}} - \mathcal{W}^* \right\|_F^2 / N$ is plotted against the normalized rank $\bar{r} = \|\boldsymbol{n}^{-1}\|_{1/2} \|\boldsymbol{r}\|_{1/2}$ (of the true tensor) defined in Equation (13). Since the choice of the regularization constant $\lambda_M$ only depends on the size of the tensor and not on the ranks of the underlying tensor in Theorem 2, we fix the regularization constant to some different values and report the dependency of the estimation error on the normalized rank $\bar{r}$ of the true tensor.

Figure 2(a) shows the result for small noise ($\sigma = 0.01$) and Figure 2(b) shows the result for large noise ($\sigma = 0.1$). As predicted by Theorem 2, the squared error $\left\| \hat{\mathcal{W}} - \mathcal{W}^* \right\|_F^2$ grows linearly against the normalized rank $\bar{r}$. This behaviour is consistently observed not only around the preferred regularization constant value (triangles) but also in the over-fitting case (circles) and the under-fitting case (crosses). Moreover, as predicted by Theorem 2, the preferred regularization constant value scales linearly and the squared error scales quadratically to the noise standard deviation $\sigma$.

As predicted by Lemma 3, the curves for the smaller $50 \times 50 \times 20$ tensor and those for the larger $100 \times 100 \times 50$ tensor seem to agree when the regularization constant is scaled by the factor two. Note that the dominant term in inequality (12) is the second term $\sqrt{\bar{n}_{\backslash k}}$, which is roughly scaled by the factor two from $50 \times 50 \times 20$ to $100 \times 100 \times 50$.

## 4.2 Tensor completion from partial observations

In this subsection, we repeat the simulation originally done by Tomioka et al. [23] and demonstrate that our results in Section 3.3 can precisely predict the empirical scaling behaviour with respect to both the size and rank of a tensor.

We present results for both matrix completion ($K = 2$) and tensor completion ($K = 3$). For the matrix case, we randomly generated low-rank matrices of dimensions $50 \times 20$, $100 \times 40$, and $250 \times 200$. For the tensor case, we randomly generated low-rank tensors of dimensions $50 \times 50 \times 20$ and $100 \times 100 \times 50$. We generated the matrices or tensors as in the previous subsection for various ranks. We randomly selected some elements of the true matrix/tensor for training and kept the

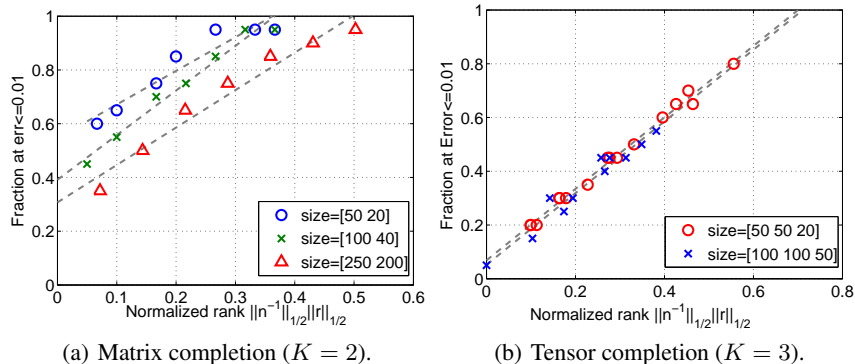

(a) Matrix completion ($K = 2$).      (b) Tensor completion ($K = 3$).

Figure 3: Scaling behaviour of matrix/tensor completion with respect to the size $\boldsymbol{n}$ and the rank $\boldsymbol{r}$.

remaining elements for testing. No observation noise is added. We used the ADMM for "as a matrix" and "constraint" approaches described in [23] to solve the minimization problem (7) for matrix completion and tensor completion, respectively. Since there is no observation noise, we chose the regularization constant $\lambda \rightarrow 0$. A single experiment for a specific size and rank can be visualized as in Figure 1.

In Figure 3, we plot the minimum fraction of observations $m = M/N$ that achieved error $\left\|\hat{\mathcal{W}} - \mathcal{W}^*\right\|_F$ smaller than $0.01$ against the normalized rank $\bar{r} = \|\boldsymbol{n}^{-1}\|_{1/2}\|\boldsymbol{r}\|_{1/2}$ (of the true tensor) defined in Equation (13). The matrix case is plotted in Figure 3(a) and the tensor case is plotted in Figure 3(b). Each series (blue crosses or red circles) corresponds to different matrix/tensor size and each data-point corresponds to a different core size (rank). We can see that the fraction of observations $m = M/N$ scales linearly against the normalized rank $\bar{r}$, which agrees with the condition $M/N \geq c_1\|\boldsymbol{n}^{-1}\|_{1/2}\|\boldsymbol{r}\|_{1/2} = c_1\bar{r}$ in Theorem 3 (see Equation (14)). The agreement is especially good for tensor completion (Figure 3(b)), where the two series almost overlap. Interestingly, we can see that when compared at the same normalized rank, tensor completion is *easier* than matrix completion. For example, when $n_k = 50$ and $r_k = 10$ for each $k = 1, \ldots, K$, the normalized rank is $0.2$. From Figure 3, we can see that we only need to see 30% of the entries in the tensor case to achieve error smaller than $0.01$, whereas we need about 60% of the entries in the matrix case.

## 5   Conclusion

We have analyzed the statistical performance of a tensor decomposition algorithm based on the overlapped Schatten 1-norm regularization (7). Numerical experiments show that our theory can predict the empirical scaling behaviour well. The fraction of observation $m = M/N$ at the threshold predicted by our theory is proportional to the quantity we call the normalized rank, which refines conjecture (sum of the mode-$k$ ranks) in [23].

There are numerous directions that the current study can be extended. In this paper, we have focused on the convergence of the estimation error; it would be meaningful to also analyze the condition for the consistency of the estimated rank as in [2]. Second, although we have succeeded in predicting the empirical scaling behaviour, the setting of random Gaussian design does not match the tensor completion setting in Section 4.2. In order to analyze the latter setting, the notion of incoherence in [5] or spikiness in [16] might be useful. This might also explain why tensor completion is easier than matrix completion at the same normalized rank. Moreover, when the target tensor is only low-rank in a certain mode, Schatten 1-norm regularization fails badly (as predicted by the high normalized rank). It would be desirable to analyze the "Mixture" approach that aims at this case [23]. In a broader context, we believe that the current paper could serve as a basis for re-examining the concept of tensor rank and low-rank approximation of tensors based on convex optimization.

**Acknowledgments.** We would like to thank Franz Király and Hiroshi Kajino for their valuable comments and discussions. This work was supported in part by MEXT KAKENHI 22700138, 23240019, 23120004, 22700289, and NTT Communication Science Laboratories.

# References

[1] E. Acar and B. Yener. Unsupervised multiway data analysis: A literature survey. *IEEE T. Knowl. Data. En.*, 21(1):6–20, 2009.

[2] F.R. Bach. Consistency of trace norm minimization. *J. Mach. Learn. Res.*, 9:1019–1048, 2008.

[3] S. Boyd and L. Vandenberghe. *Convex Optimization*. Cambridge University Press, 2004.

[4] R. Bro. PARAFAC. Tutorial and applications. *Chemometr. Intell. Lab.*, 38(2):149–171, 1997.

[5] E. J. Candes and B. Recht. Exact matrix completion via convex optimization. *Found. Comput. Math.*, 9(6):717–772, 2009.

[6] J.D. Carroll and J.J. Chang. Analysis of individual differences in multidimensional scaling via an n-way generalization of "Eckart-Young" decomposition. *Psychometrika*, 35(3):283–319, 1970.

[7] P. Comon. Tensor decompositions. In J. G. McWhirter and I. K. Proudler, editors, *Mathematics in signal processing V*. Oxford University Press, 2002.

[8] L. De Lathauwer and J. Vandewalle. Dimensionality reduction in higher-order signal processing and rank-$(r_1, r_2, \ldots, r_n)$ reduction in multilinear algebra. *Linear Algebra Appl.*, 391:31–55, 2004.

[9] K. Fukumizu. Generalization error of linear neural networks in unidentifiable cases. In *Algorithmic Learning Theory*, pages 51–62. Springer, 1999.

[10] S. Gandy, B. Recht, and I. Yamada. Tensor completion and low-n-rank tensor recovery via convex optimization. *Inverse Problems*, 27:025010, 2011.

[11] J. Håstad. Tensor rank is NP-complete. *Journal of Algorithms*, 11(4):644–654, 1990.

[12] T. G. Kolda and B. W. Bader. Tensor decompositions and applications. *SIAM Review*, 51(3):455–500, 2009.

[13] J. Liu, P. Musialski, P. Wonka, and J. Ye. Tensor completion for estimating missing values in visual data. In *Prof. ICCV*, 2009.

[14] M. Mørup. Applications of tensor (multiway array) factorizations and decompositions in data mining. *Wiley Interdisciplinary Reviews: Data Mining and Knowledge Discovery*, 1(1):24–40, 2011.

[15] S. Negahban, P. Ravikumar, M. Wainwright, and B. Yu. A unified framework for high-dimensional analysis of $m$-estimators with decomposable regularizers. In Y. Bengio, D. Schuurmans, J. Lafferty, C. K. I. Williams, and A. Culotta, editors, *Advances in NIPS 22*, pages 1348–1356. 2009.

[16] S. Negahban and M.J. Wainwright. Restricted strong convexity and weighted matrix completion: Optimal bounds with noise. Technical report, arXiv:1009.2118, 2010.

[17] S. Negahban and M.J. Wainwright. Estimation of (near) low-rank matrices with noise and high-dimensional scaling. *Ann. Statist.*, 39(2), 2011.

[18] B. Recht, M. Fazel, and P.A. Parrilo. Guaranteed minimum-rank solutions of linear matrix equations via nuclear norm minimization. *SIAM Review*, 52(3):471–501, 2010.

[19] A. Rohde and A.B. Tsybakov. Estimation of high-dimensional low-rank matrices. *Ann. Statist.*, 39(2):887–930, 2011.

[20] N.D. Sidiropoulos, R. Bro, and G.B. Giannakis. Parallel factor analysis in sensor array processing. *IEEE T. Signal Proces.*, 48(8):2377–2388, 2000.

[21] M. Signoretto, L. De Lathauwer, and J.A.K. Suykens. Nuclear norms for tensors and their use for convex multilinear estimation. Technical Report 10-186, ESAT-SISTA, K.U.Leuven, 2010.

[22] N. Srebro, J. D. M. Rennie, and T. S. Jaakkola. Maximum-margin matrix factorization. In Lawrence K. Saul, Yair Weiss, and Léon Bottou, editors, *Advances in NIPS 17*, pages 1329–1336. MIT Press, Cambridge, MA, 2005.

[23] R. Tomioka, K. Hayashi, and H. Kashima. Estimation of low-rank tensors via convex optimization. Technical report, arXiv:1010.0789, 2011.

[24] L. R. Tucker. Some mathematical notes on three-mode factor analysis. *Psychometrika*, 31(3):279–311, 1966.

[25] M. Vasilescu and D. Terzopoulos. Multilinear analysis of image ensembles: Tensorfaces. *Computer Vision—ECCV 2002*, pages 447–460, 2002.

[26] H. Wang and N. Ahuja. Facial expression decomposition. In *Proc. 9th ICCV*, pages 958 – 965, 2003.

